# Fusion of Similarity Data in Clustering

**Tilman Lange and Joachim M. Buhmann**
(langet,jbuhmann)@inf.ethz.ch
Institute of Computational Science, Dept. of Computer Sience,
ETH Zurich, Switzerland

## Abstract

Fusing multiple information sources can yield significant benefits to successfully accomplish learning tasks. Many studies have focussed on fusing information in *supervised* learning contexts. We present an approach to utilize multiple information sources in the form of similarity data for *unsupervised* learning. Based on similarity information, the clustering task is phrased as a non-negative matrix factorization problem of a mixture of similarity measurements. The tradeoff between the informativeness of data sources and the sparseness of their mixture is controlled by an entropy-based weighting mechanism. For the purpose of model selection, a stability-based approach is employed to ensure the selection of the most self-consistent hypothesis. The experiments demonstrate the performance of the method on toy as well as real world data sets.

## 1 Introduction

*Clustering* has found increasing attention in the past few years due to the enormous information flood in many areas of information processing and data analysis. The ability of an algorithm to determine an interesting partition of the set of objects under consideration, however, heavily depends on the available information. It is, therefore, reasonable to equip an algorithm with as much information as possible and to endow it with the capability to distinguish between *relevant* and *irrelevant* information sources. How to reasonably identify a weighting of the different information sources such that an interesting group structure can be successfully uncovered, remains, however, a largely unresolved issue.

Different sources of information about the same objects naturally arise in many application scenarios. In computer vision, for example, information sources can consist of plain intensity measurements, edge maps, the similarity to other images or even human similarity assessments. Similarly in bio-informatics: the similarity of proteins,e.g., can be assessed in different ways, ranging from the comparison of gene profiles to direct comparisons at the sequence level using alignment methods.

In this work, we use a non-negative matrix factorization approach (nmf) to *pairwise* clustering of similarity data that is extended in a second step in order to incorporate a suitable weighting of multiple information sources, leading to a *mixture* of similarities. The latter represents the main contribution of this work. Algorithms for nmf have recently found a lot of attention. Our proposal is inspired by the work in [11] and [5]. Only recently, [18] have also employed a nmf to perform clustering. For the purpose of model selection, we employ a stability-based approach that has already been successfully applied to model se-

lection problems in clustering (e.g. in [9]). Instead of following the strategy to first embed the similarities into a space with Euclidean geometry and then to perform clustering and, where required, feature selection/weighting on the stacked feature vector, we advocate an approach that is closer to the original similarity data by performing nmf.

Some work has been devoted to feature selection and weighting in clustering problems. In [13] a variant of the $k$-means algorithm has been studied that employs the Fisher criterion to assess the importance of individual features. In [14, 10], Gaussian mixture model-based approaches to feature selection are introduced. The more general problem of learning a suitable metric has also been investigated, e.g. in [17]. Similarity measurements represent a particularly generic form of providing input to a clustering algorithm. Fusing such representations has only recently been studied in the context of kernel-based *supervised* learning, e.g. in [7] using semi-definite programming and in [3] using a boosting procedure. In [1], an approach to learning the bandwidth parameter of an rbf-kernel for spectral clustering is studied.

The paper is organized as follows: section 2 introduces the nmf-based clustering method combined with a data-source weighting (section 3). Section 4 discusses an out-of-sample extension allowing us to *predict* assignments and to employ the *stability principle* for model selection. Experimental evidence in favor of our approach is given in section 5.

## 2 Clustering by Non-Negative Matrix Factorization

Suppose we want to group a finite set of objects $\mathcal{O}_n := \{o_1, \ldots, o_n\}$. Usually, there are multiple ways of measuring the similarity between different objects. Such relations give rise to similarities $s_{ij} := s(o_i, o_j)$ [1] where we assume non-negativity $s_{ij} \geq 0$, symmetry $s_{ji} = s_{ij}$, and boundedness $s_{ij} < \infty$. For $n$ objects, we summarize the similarity data in a $n \times n$ matrix $\mathbf{S} = (s_{ij})$ which is re-normalized to $\mathbf{P} = \mathbf{S}/\mathbf{1}_n^t \mathbf{S} \mathbf{1}_n$, where $\mathbf{1}_n := (1, \ldots, 1)^t$. The re-normalized similarities can be interpreted as the probability of the *joint* occurrence of objects $i, j$.

We aim now at finding a non-negative matrix factorization of $\mathbf{P} \in [0,1]^{n \times n}$ into a product $\mathbf{W}\mathbf{H}^t$ of the $n \times k$ matrices $\mathbf{W}$ and $\mathbf{H}$ with non-negative entries for which additionally holds $\mathbf{1}_n^t \mathbf{W} \mathbf{1}_k = 1$ and $\mathbf{H}^t \mathbf{1}_n = \mathbf{1}_k$, where $k$ denotes the number of clusters. That is, one aims at explaining the overall probability for a co-occurrence by a latent cause, the unobserved classes. The constraints ensure, that the entries of both, $\mathbf{W}$ and $\mathbf{H}$, can be considered as probabilities: the entry $w_{i\nu}$ of $\mathbf{W}$ is the joint probability $q(i, \nu)$ of object $i$ and class $\nu$ whereas $h_{jk}$ in $\mathbf{H}$ is the probability $q(j|\nu)$. This model implicitly assumes independence of object $i$ and $j$ conditioned on $\nu$. Given a factorization of $\mathbf{P}$ in $\mathbf{W}$ and $\mathbf{H}$, we can use the maximum a posteriori estimate, $\arg\max_\nu h_{i\nu} \sum_j w_{j\nu}$, to arrive at a *hard* assignment of objects to classes.

In order to obtain a factorization, we *minimize the cross-entropy*

$$C(\mathbf{P} \| \mathbf{W}\mathbf{H}^t) := -\sum_{i,j} p_{ij} \log \sum_\nu w_{i\nu} h_{j\nu} \tag{1}$$

which becomes minimal *iff* $\mathbf{P} = \mathbf{W}\mathbf{H}^t$ [2] and is not convex in $\mathbf{W}$ and $\mathbf{H}$ together. Note, that the factorization is not necessarily unique. We resort to a local optimization scheme, which is inspired by the Expectation-Maximization (EM) algorithm: Let $\tau_{\nu ij} \geq 0$ with $\sum_\nu \tau_{\nu ij} = 1$. Then, by the convexity of $-\log x$, we obtain $-\log \sum_\nu w_{i\nu} h_{j\nu} \leq -\sum_\nu \tau_{\nu ij} \log \frac{w_{i\nu} h_{j\nu}}{\tau_{\nu ij}}$,

which yields the relaxed objective function:

$$\tilde{C}(\mathbf{P}\|\mathbf{W}\mathbf{H}^t) := -\sum_{i,j,\nu} p_{ij}\tau_{\nu ij}\log w_{i\nu}h_{j\nu} + \tau_{\nu ij}\log\tau_{\nu ij} \geq C(\mathbf{P}\|\mathbf{W}\mathbf{H}^t). \qquad (2)$$

With this relaxation, we can employ an *alternating minimization* scheme for minimizing the bound on $C$. As in EM, one iterates

1. Given $\mathbf{W}$ and $\mathbf{H}$, minimize $\tilde{C}$ w.r.t. $\tau_{\nu ij}$

2. Given the values $\tau_{\nu ij}$, find estimates for $\mathbf{W}$ and $\mathbf{H}$ by minimizing $\tilde{C}$.

until convergence, which produces a sequence of estimates

$$\tau_{\nu ij}^{(t)} = \frac{w_{i\nu}^{(t)}h_{j\nu}^{(t)}}{\sum_\mu w_{i\mu}^{(t)}h_{j\mu}^{(t)}}, \qquad w_{i\nu}^{(t+1)} = \sum_j p_{ij}\tau_{\nu ij}^{(t)}, \qquad h_{j\nu}^{(t+1)} = \frac{\sum_i p_{ij}\tau_{\nu ij}^{(t)}}{\sum_{a,b} p_{ab}\tau_{\nu ab}^{(t)}} \qquad (3)$$

that converges to a local minimum of $\tilde{C}$. This is an instance of an MM algorithm [8]. We use the convention $h_{j\nu} = 0$ whenever $\sum_{i,j} p_{ij}\tau_{\nu ij} = 0$. The per-iteration complexity is $O(n^2)$.

## 3 Fusing Multiple Data Sources

Measuring the similarity of objects in, say, $L$ different ways results in $L$ normalized similarity matrices $\mathbf{P}_1, \ldots, \mathbf{P}_L$. We introduce now weights $\alpha_l, 1 \leq l \leq L$, with $\sum_l \alpha_l = 1$. For fixed $\boldsymbol{\alpha} = (\alpha_l) \in [0,1]^L$, the aggregated and normalized similarity becomes the convex combination $\bar{\mathbf{P}} = \sum_l \alpha_l \mathbf{P}_l$. Hence, $\bar{p}_{ij}$ is a *mixture* of individual similarities $p_{ij}^{(l)}$, i.e. a mixture of different explanations. Again, we seek a good factorization of $\bar{\mathbf{P}}$ by minimizing the cross-entropy, which then becomes

$$\min_{\boldsymbol{\alpha},\mathbf{W},\mathbf{H}} \mathbb{E}_{\boldsymbol{\alpha}}\left[C(\mathbf{P}_l\|\mathbf{W}\mathbf{H}^t)\right] \qquad (4)$$

where $\mathbb{E}_{\boldsymbol{\alpha}}[f_l] = \sum_l \alpha_l f_l$ denotes the expectation w.r.t. the discrete distribution $\boldsymbol{\alpha}$. The same relaxation as in the last section can be used, i.e. for all $\boldsymbol{\alpha}$, $\mathbf{W}$ and $\mathbf{H}$, we have $\mathbb{E}_{\boldsymbol{\alpha}}[C(\mathbf{P}_l\|\mathbf{W}\mathbf{H}^t)] \leq \mathbb{E}_{\boldsymbol{\alpha}}[\tilde{C}(\mathbf{P}_l\|\mathbf{W}\mathbf{H}^t)]$. Hence, we can employ a slightly modified, *nested* alternating minimization approach: Given fixed $\boldsymbol{\alpha}$, obtain estimates $\mathbf{W}$ and $\mathbf{H}$ using the relaxation of the last section. The update equations change to

$$w_{i\nu}^{(t+1)} = \sum_l \alpha_l \sum_j p_{ij}^{(l)}\tau_{\nu ij}^{(t)}, \qquad h_{j\nu}^{(t+1)} = \frac{\sum_l \alpha_l \sum_i p_{ij}^{(l)}\tau_{\nu ij}^{(t)}}{\sum_l \alpha_l \sum_{i,j} p_{ij}^{(l)}\tau_{\nu ij}^{(t)}}. \qquad (5)$$

Given the current estimates of $\mathbf{W}$ and $\mathbf{H}$, we could minimize the objective in equation (4) w.r.t. $\boldsymbol{\alpha}$ subject to the constraint $\|\boldsymbol{\alpha}\|_1 = 1$. To this end, set $c_l := C(\mathbf{P}_l\|\mathbf{W}\mathbf{H}^t)$ and let $\mathbf{c} = (c_l)_l$. Minimizing the expression in equation (4) subject to the constraints $\sum_l \alpha_l = 1$ and $\boldsymbol{\alpha} \succeq 0$, therefore, becomes a *linear program (LP)* $\min_{\boldsymbol{\alpha}} \mathbf{c}^t\boldsymbol{\alpha}$ such that $\mathbf{1}_L^t\boldsymbol{\alpha} = 1$, $\boldsymbol{\alpha} \succeq 0$, where $\succeq$ denotes the element-wise $\geq$-relation. The LP solution is very sparse since the optimal solutions for the linear program lie on the corners of the simplex in the positive orthant spanned by the constraints. In particular, it lacks a means to control the *sparseness* of the coefficients $\boldsymbol{\alpha}$. We, therefore, use a maximum entropy approach ([6]) for sparseness control: the entropy is upper bounded by $\log L$ and measures the sparseness of the vector $\boldsymbol{\alpha}$, since the lower the entropy the more peaked the distribution $\boldsymbol{\alpha}$ can be. Hence, by *lower bounding* the entropy, we specify the maximal admissible sparseness. This approach is reasonable as we actually want to combine *multiple* (not only identify one) information sources but the best fit in an unsupervised problem will be usually obtained by choosing only a single

source. Thus, we modify the objective originally given in eq. (4) to the entropy-regularized problem $\mathbb{E}_{\boldsymbol{\alpha}}[\tilde{C}(\mathbf{P}_l \| \mathbf{W}\mathbf{H}^t)] - \eta H(\boldsymbol{\alpha})$, so that the mathematical program given above becomes

$$\min_{\boldsymbol{\alpha}} \mathbf{c}^t\boldsymbol{\alpha} - \eta H(\boldsymbol{\alpha}) \qquad \text{s.t. } \mathbf{1}_L^t \boldsymbol{\alpha} = 1, \; \boldsymbol{\alpha} \succeq 0, \tag{6}$$

where $H$ denotes the (discrete) entropy and $\eta \in \mathbb{R}_+$ is a positive Lagrange parameter. The optimization problem in eq. (6) has an analytical solution, namely the Gibbs distribution

$$\alpha_l \propto \exp(-c_l/\eta) \tag{7}$$

For $\eta \to \infty$ one obtains $\alpha_l = 1/L$, while for $\eta \to 0$, the LP solution is recovered and the estimates become the sparser the more the individual $c_l$ differ. Put differently, the parameter $\eta$ enables us to explore the space of different similarity combinations. The issue of selecting a reasonable value for the parameter $\eta$ will be discussed in the next section.

Iterating this nested procedure will yield a *locally* optimal solution to the problem of minimizing the entropy-constrained objective, since (i) we obtain a local minimum of the modified objective function and (ii) solving the outer optimization problem can only further *decrease* the entropy-constrained objective function.

## 4 Generalization and Model Selection

In this section, we introduce an out-of-sample extension that allows us to classify objects, that have not been used for learning the parameters $\boldsymbol{\alpha}$, $\mathbf{W}$ and $\mathbf{H}$. The extension mechanism can be seen as in spirit of the Nyström extension (c.f. [16]). Introducing such a generalization mechanism is worthwhile for two reasons: (i) To speed-up the computation if the number $n$ of objects under consideration is very large: By selecting a small subset of $m \ll n$ objects for the initial fit followed by the application of a computationally less expensive prediction step, one can realize such a speed-up. (ii) The free parameters of the approach, the number of clusters $k$ as well as the sparseness control parameter $\eta$, can be estimated using a re-sampling-based stability assessment that relies on the ability of an algorithm to generalize to previously unseen objects.

**Out-of-Sample Extension:** Suppose we have to predict class memberships for $r$ $(= n - m$ in the hold-out case) additional objects in the $r \times m$ matrix $\tilde{\mathbf{S}}_l$. Given the decomposition into $\mathbf{W}$ and $\mathbf{H}$, let $z_{ik}$ be the "posterior" estimated for the $i$-th object in the data set used for the original fit, i.e. $z_{i\nu} \propto h_{i\nu} \sum_j w_{j\nu}$. We can express the weighted, normalized similarity between a new object $o$ and object $i$ as $\hat{p}_{io} := \sum_l \alpha_l \tilde{s}_{oi}^{(l)} / \sum_{l,j} \alpha_l \tilde{s}_{oj}^{(l)}$. We *approximate* now $z_{o\nu}$ for a new object $o$ by

$$\hat{z}_{o\nu} = \sum_i z_{i\nu} \hat{p}_{io}, \tag{8}$$

which amounts to an *interpolation* of the $z_{o\nu}$. These values can be obtained using the originally computed $z_{i\nu}$ which are weighted according to their similarity between object $i$ and $o$. In the analogy to the Nyström approximation, the $(z_{i\nu})$ play the role of basis elements while the $\hat{p}_{io}$ amount to coefficients in the basis approximation. The prediction procedure requires $O(mr(l + r + k))$ steps.

**Model Selection:** The approach presented so far has two free parameters, the number of classes $k$ and the sparseness penalty $\eta$. In [9], a method for determining the number of classes has been introduced, that assesses the variability of clustering solutions. Thus, we focus on selecting $\eta$ using stability. The assessment can be regarded as a generalization of cross-validation, as it relies on the dissimilarity of solutions generated from multiple sub-samples. In a second step, the solutions obtained from these samples are extended to the complete data set by an appropriate predictor. Multiple classifications of the same data

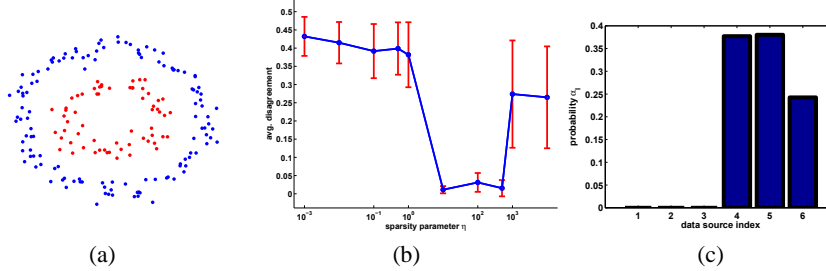

<div align="center">

(a)            (b)            (c)

</div>

Figure 1: Results on the toy data set (1(a)): The stability assessment (1(b)) suggests the range $\eta \in \{10^1, 10^2, 5 \cdot 10^2\}$, which yield solutions matching the ground-truth. In 1(c), the $\alpha_l$ are depicted for a sub-sample and $\eta$ in this range.

set are obtained, whose similarity can be measured. For two clustering solutions $\mathbf{Y}, \mathbf{Y}' \in \{1, \ldots, k\}^n$, we define their disagreement as

$$d(\mathbf{Y}, \mathbf{Y}') = \min_{\pi \in \mathfrak{S}_k} \frac{1}{n} \sum_{i=1}^{n} \mathbb{I}_{\{y_i \neq \pi(y'_i)\}} \tag{9}$$

where $\mathfrak{S}_k$ denotes the set of all permutation on sets of size $k$ and $\mathbb{I}_A$ is the indicator function on the expression $A$. The measure quantifies the 0-1 loss after the labels have been permuted, so that the two clustering solutions are in the best possible agreement. Perfect agreement up to a permutation of the labels implies $d(\mathbf{Y}, \mathbf{Y}') = 0$. The optimal permutation can be determined in $O(k^3)$ by phrasing the problem as a weighted bipartite matching problem. Following the approach in [9], we select the $\eta$, given a pre-specified range of admissible values, such that the *average* disagreement observed on $B$ sub-samples is minimal. In this sense, the entropy regularization mechanism guides the search for similarity combinations leading to stable grouping solutions. Note that, multiple minima can occur and may yield solutions emphasizing different aspects of the data.

## 5 Experimental Results and Discussion

The performance of our proposal is explored by analyzing toy and real world data. For the model selection (sec. 4), we have used $B = 20$ sub-samples with the proposed out-of-sample extension for prediction. For the stability assessment, different $\eta$ have been chosen by $\eta \in \{10^{-3}, 10^{-2}, 10^{-1}, .5, 1, 10^1, 10^2, 5 \cdot 10^2, 10^3, 10^4\}$. We compared our results with NCut [15] and Lee and Seung's two NMF algorithms [11] (which measure the approximation error of the factorization with (i) the KL divergence and (ii) the squared Frobenius norm) applied to the uniform combination of similarities.

**Toy Experiment:** Figure 1(a) depicts a data set consisting of two nested rings, where the clustering task consists of identifying each ring as a class. We used rbf-kernels $k(\mathbf{x}, \mathbf{y}) = \exp(-\|\mathbf{x} - \mathbf{y}\|^2 / 2\sigma^2)$ for $\sigma$ varying in $\{10^{-4}, 10^{-3}, 10^{-2}, 10^0, 10^1\}$ as well as the path kernel introduced in [4]. All methods *fail* when used with the individual kernels except for the path-kernel. The non-trivial problem is to detect the correct structure despite the disturbing influence of 5 un-informative kernels. Data sets of size $\lceil n/5 \rceil$ have been generated by sub-sampling. Figure 1(b) depicts the stability assessment, where we see very small disagreements for $\eta \in \{10^1, 10^2, 5 \cdot 10^2\}$. At the minimum, the solution almost perfectly matches the ground-truth (1 error). A plot of the resulting $\boldsymbol{\alpha}$-coefficients is given in figure 1(c). NCut as well as the other nmf-methods lead to an error rate of $\approx 0.5$ when applied to the uniformly combined similarities.

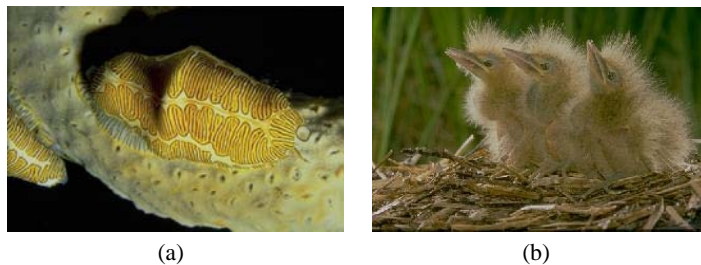

(a)                                                    (b)

Figure 2: Images for the segmentation experiments.

**Image segmentation example:**[3]    The next task consists of finding a reasonable segmentation of the images depicted in figures 2(b) and 2(a). For both images, we measured localized intensity histograms and additionally computed Gabor filter responses (e.g. [12]) on 3 scales for 4 different orientations. For each response image, the same histogramming procedure has been used. For all the histograms, we computed the pairwise Jensen-Shannon divergence (e.g. [2]) for all pairs $(i, j)$ of image sites and took the element-wise exponential of the negative Jensen-Shannon divergences. The resulting similarity matrices have been used as input for the nmf-based data fusion. For the sub-sampling, $m = 500$ objects have been employed. Figures 3(a) (for the shell image) and 3(b) (for the bird image) show the stability curves for these examples which exhibit minima for non-trivial $\eta$ resulting in non-uniform $\boldsymbol{\alpha}$. Figure 3(c) depicts the resulting segmentation generated using $\boldsymbol{\alpha}$ indicated by the stability assessment, while 3(d) shows a segmentation result, where $\boldsymbol{\alpha}$ is closer to the uniform distribution but the stability score for the corresponding $\eta$ is low. Again, we can see that weighting the different similarity measurements has a beneficial effect, since it leads to improved results. The comparison with the NCut result on the uniformly weighted data (fig. 3(e)) confirms that a non-trivial weighting is desirable here. Note that we have used the full data set with NCut. For, the image in fig. 2(b), we observe similar behavior: the stability selected solution (fig. 3(f)) is more meaningful than the NCut solution (fig. 3(g)) obtained on the uniformly weighted data. In this example, the intensity information dominates the solution obtained on the uniformly combined similarities. However, the texture information alone does *not* yield a sensible segmentation. Only the non-trivial combination, where the influence of intensity information is decreased and that of the texture information is increased, gives rise to the desired result. It is additionally noteworthy, that the prediction mechanism employed works rather well: In both examples, it has been able to generalize the segmentation from $m = 500$ to more than 3500 objects. However, artifacts resulting from the subsampling-and-prediction procedure cannot always be avoided, as can be seen in 3(f). They vanish, however, once the algorithm is re-applied to the full data (fig. 3(h)).

**Clustering of Protein Sequences:**    Our final application is about the functional categorization of yeast proteins. We partially adopted the data used in [7] [4]. Since several of the 3588 proteins belong to *more than one* category, we extracted a subset of 1579 proteins exclusively belonging to one of the three categories *cell cycle + DNA processing*,*transcription* and *protein fate*. This step ensures a clear ground-truth for comparison. Of the matrices used in [7], we employed a Gauss Kernel derived from gene expression profiles, one derived from Swiss-Waterman alignments, one obtained from comparisons of protein domains as well as two diffusion kernels derived from protein-protein interaction data. Although the data is not very discriminative for the 3-class problem, the solutions generated on the data combined using the $\boldsymbol{\alpha}$ for the most stable $\eta$ lead to more than $10\%$ improvement w.r.t. the

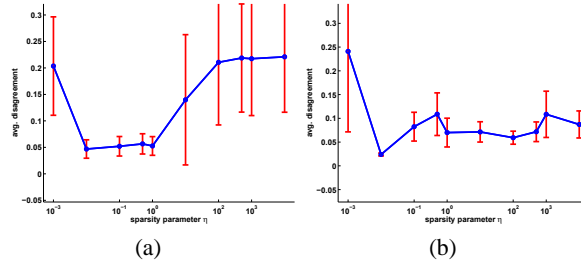

(a)  (b)

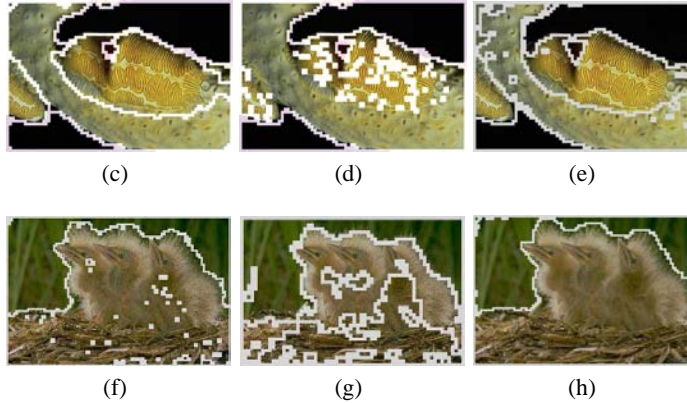

(c)  (d)  (e)

(f)  (g)  (h)

Figure 3: Stability plots and segmentation results for the images in 2(a) and 2(b) (see text).

ground-truth (the disagreement measure of section 4 is used) in comparison with the solution obtained using the least stable $\eta$-parameter. The latter, however, was hardly better than random guessing by having an overall disagreement of more than $0.60$ (more precisely, $0.6392 \pm 0.0455$) on this data. For the most stable $\eta$, we observed a disagreement around $0.52$ depending on the sub-sample (best $0.5267 \pm 0.0403$). In this case, the largest weight was assigned to the protein-protein interaction data. NCut and the two nmf methods proposed in [11] lead to rates $0.5953$, $0.6080$ and $0.6035$, respectively, when applied to the naive combination. Note, that the clustering results are comparable with some of those obtained in [7], where the protein-protein interaction data has been used to construct a (supervised) classifier.

## 6    Conclusion

This work introduced an approach to combining similarity data originating from multiple sources for grouping a set of objects. Adopting a pairwise clustering perspective enables a smooth integration of multiple similarity measurements. To be able to distinguish between desired and distractive information, a weighting mechanism is introduced leading to a potentially sparse convex combination of the measurements. Here, an entropy constraint is employed to control the amount of sparseness actually allowed. A stability-based model selection mechanism is used to select this free parameter. We emphasize, that this procedure represents a completely unsupervised model selection strategy. The experimental evaluation on toy and real world data demonstrates that our proposal yields meaningful partitions and is able to distinguish between desired and spurious structure in data.

Future work will focus on (i) improving the optimization of the proposed model, (ii) the

integration of additional constraints and (iii) the introduction of a *cluster-specific* weighting mechanism. The proposed method as well as its relation to other approaches discussed in the literature is currently under further investigation.

## Footnotes

[1] In the following, we represent objects by their indices.

[2] The Kullback-Leibler divergence is $D(\mathbf{P} \| \mathbf{W}\mathbf{H}^t) = -H(\mathbf{P}) + C(\mathbf{P} \| \mathbf{W}\mathbf{H}^t) \geq 0$ with equality *iff* $\mathbf{P} = \mathbf{W}\mathbf{H}^t$.

[3]Only comparisons with NCut reported. The nmf results are slightly worse than those of NCut.

[4]The data is available at `http://noble.gs.washington.edu/proj/yeast/`.

# References

[1] F. R. Bach and M. I. Jordan. Learning spectral clustering. In *NIPS*, volume 16. MIT Press, 2004.

[2] J. Burbea and C. R. Rao. On the convexity of some divergence measures based on entropy functions. *IEEE Trans. Inform. Theory*, 28(3), 1982.

[3] K. Crammer, J. Keshet, and Y. Singer. Kernel design using boosting. In *NIPS*, volume 15. MIT Press, 2003.

[4] B. Fischer, V. Roth, and J. M. Buhmann. Clustering with the connectivity kernel. In *NIPS*, volume 16. MIT Press, 2004.

[5] Thomas Hofmann. Unsupervised learning by probabilistic latent semantic analysis. *Mach. Learn.*, 42(1-2):177–196, 2001.

[6] E. T. Jaynes. Information theory and statistical mechanics, I and II. *Physical Reviews*, 106 and 108:620–630 and 171–190, 1957.

[7] G. R. G. Lanckriet, M. Deng, N. Cristianini, M. I. Jordan, and W. S. Noble. Kernel-based data fusion and its application to protein function prediction in yeast. In *Pacific Symposium on Biocomputing*, pages 300–311, 2004.

[8] Kenneth Lange. *Optimization*. Springer Texts in Statistics. Springer, 2004.

[9] T. Lange, M. Braun, V. Roth, and J.M. Buhmann. Stability-based model selection. In *NIPS*, volume 15. MIT Press, 2003.

[10] M. H. C. Law, M. A. T. Figueiredo, and A. K. Jain. Simultaneous feature selection and clustering using mixture models. *IEEE Trans. Pattern Anal. Mach. Intell.*, 26(9):1154–1166, 2004.

[11] Daniel D. Lee and H. Sebastian Seung. Algorithms for non-negative matrix factorization. In *NIPS*, volume 13, pages 556–562, 2000.

[12] B. S. Manjunath and W. Y. Ma. Texture features for browsing and retrieval of image data. *IEEE Trans. Pattern Anal. Mach. Intell.*, 18(8):837–842, 1996.

[13] D. S. Modha and W. S. Spangler. Feature weighting in k-means clustering. *Mach. Learn.*, 52(3):217–237, 2003.

[14] V. Roth and T. Lange. Feature selection in clustering problems. In *NIPS*, volume 16. MIT Press, 2004.

[15] Jianbo Shi and Jitendra Malik. Normalized cuts and image segmentation. *IEEE Trans. Pattern Anal. Mach. Intell.*, 22(8):888–905, 2000.

[16] C. K. I. Williams and M. Seeger. Using the Nystrï¿$\frac{1}{2}$m method to speed up kernel machines. In *NIPS*, volume 13. MIT Press, 2001.

[17] E. Xing, A. Ng, M. Jordan, and S. Russell. Distance metric learning with application to clustering with side-information. In *NIPS*, volume 15, 2003.

[18] W. Xu, X. Liu, and Y. Gong. Document clustering based on non-negative matrix factorization. In *SIGIR '03*, pages 267–273. ACM Press, 2003.
